# Learning Efficient Markov Networks

**Vibhav Gogate**  **William Austin Webb**  **Pedro Domingos**
Department of Computer Science & Engineering
University of Washington
Seattle, WA 98195. USA
{vgogate,webb,pedrod}@cs.washington.edu

## Abstract

We present an algorithm for learning high-treewidth Markov networks where inference is still tractable. This is made possible by exploiting context-specific independence and determinism in the domain. The class of models our algorithm can learn has the same desirable properties as thin junction trees: polynomial inference, closed-form weight learning, etc., but is much broader. Our algorithm searches for a feature that divides the state space into subspaces where the remaining variables decompose into independent subsets (conditioned on the feature and its negation) and recurses on each subspace/subset of variables until no useful new features can be found. We provide probabilistic performance guarantees for our algorithm under the assumption that the maximum feature length is bounded by a constant $k$ (the treewidth can be much larger) and dependences are of bounded strength. We also propose a greedy version of the algorithm that, while forgoing these guarantees, is much more efficient. Experiments on a variety of domains show that our approach outperforms many state-of-the-art Markov network structure learners.

## 1  Introduction

Markov networks (also known as Markov random fields, etc.) are an attractive class of joint probability models because of their generality and flexibility. However, this generality comes at a cost. Inference in Markov networks is intractable [25], and approximate inference schemes can be unreliable, and often require much hand-crafting. Weight learning has no closed-form solution, and requires convex optimization. Computing the gradient for optimization in turn requires inference. Structure learning – the problem of finding the features of the Markov network – is also intractable [15], and has weight learning and inference as subroutines.

Intractable inference and weight optimization can be avoided if we restrict ourselves to *decomposable* Markov networks [22]. A decomposable model can be expressed as a product of distributions over the cliques in the graph divided by the product of the distributions of their intersections. An arbitrary Markov network can be converted into a decomposable one by triangulation (adding edges until every cycle of length four or more has at least one chord). The resulting structure is called a *junction tree*. Goldman [13] proposed a method for learning Markov networks without numeric optimization based on this idea. Unfortunately, the triangulated network can be exponentially larger than the original one, limiting the applicability of this method. More recently, a series of papers have proposed methods for directly learning junction trees of bounded treewidth ([2, 21, 8] etc.). Unfortunately, since the complexity of inference (and typically of learning) is exponential in the treewidth, only models of very low treewidth (typically 2 or 3) are feasible in practice, and thin junction trees have not found wide applicability.

Fortunately, low treewidth is an overly strong condition. Models can have high treewidth and still allow tractable inference and closed-form weight learning from a reasonable number of samples, by exploiting context-specific independence [6] and determinism [7]. Both of these result in clique dis-

tributions that can be compactly expressed even if the cliques are large. In this paper we propose a learning algorithm based on this observation. Inference algorithms that exploit context-specific independence and determinism [7, 26, 11] have a common structure: they search for partial assignments to variables that decompose the remaining variables into independent subsets, and recurse on these smaller problems until trivial ones are obtained. Our algorithm uses a similar strategy, but at learning time: it recursively attempts to find features (i.e., partial variable assignments) that decompose the problem into smaller (nearly) independent subproblems, and stops when the data does not warrant further decomposition.

Decomposable models can be expressed as both Markov networks and Bayesian networks, and state-of-the-art Bayesian network learners extensively exploit context-specific independence [9]. However, they typically still learn intractable models. Lowd and Domingos [18] learned tractable high-treewidth Bayesian networks by penalizing inference complexity along with model complexity in a standard Bayesian network learner. Our approach can learn exponentially more compact models by exploiting the additional flexibility of Markov networks, where features can overlap in arbitrary ways. It can greatly speed up learning relative to standard Markov network learners because it avoids weight optimization and inference, while Lowd and Domingos' algorithm is much slower than standard Bayesian network learning (where, given complete data, weight optimization and inference are already unnecessary). Perhaps most significantly, it is also more fundamental in that it is based on identifying what makes inference tractable and directly exploiting it, potentially leading to a much better accuracy/inference cost trade-off. As a result, our approach has formal guarantees, which Lowd and Domingos' algorithm lacks.

We provide both theoretical guarantees and empirical evidence for our approach. First, we provide probabilistic performance guarantees for our algorithm by making certain assumptions about the underlying distribution. These results rely on exhaustive search over features up to length $k$. (The treewidth of the resulting model can still be as large as the number of variables.) We then propose greedy heuristics for more efficient learning, and show empirically that the Markov networks learned in this way are more accurate than thin junction trees as well as networks learned using the algorithm of Della Pietra et al. [12] and L1 regularization [16, 24], while allowing much faster inference (which in practice translates into more accurate query answers).

## 2 Background: Junction Trees and Feature Graphs

We denote sets by capital letters and members of a set by small letters. A double capital letter denotes a set of subsets. We assume that all random variables have binary domains $\{0,1\}$ (or $\{$false,true$\}$). We make this assumption for simplicity of exposition; our analysis extends trivially to multi-valued variables.

We begin with some necessary definitions. An atomic feature or literal is an assignment of a value to a variable. $x$ denotes the assignment $x = 1$ while $\neg x$ denotes $x = 0$ (note that the distinction between an atomic feature $x$ and the variable which is also denoted by $x$ is usually clear from context). A feature, denoted by $F$, defined over a subset of variables $V(F)$ is formed by conjoining atomic features or literals, e.g., $x_1 \wedge \neg x_2$ is a feature formed by conjoining two atomic features $x_1$ and $\neg x_2$. Given an assignment, denoted by $\overline{V(F)}$, to all variables of $F$, $F$ is said to be satisfied or assigned the value 1 iff for all literals $l \in F$, it also holds that $l \in \overline{V(F)}$. A feature that is not satisfied is said to be assigned the value 0. Often, given a feature $F$, we will abuse notation and write $\overline{V(F)}$ as $\overline{F}$.

A Markov network or a log-linear model is defined as a set of pairs $(F_i, w_i)$ where $F_i$ is a feature and $w_i$ is its weight. It represents the following joint probability distribution:

$$P(\overline{V}) = \frac{1}{Z} \exp\left( \sum_i w_i \times F_i(\overline{V}_{V(F_i)}) \right) \quad (1)$$

where $\overline{V}$ is a truth-assignment to all variables $V = \cup_i V(F_i)$, $F_i(\overline{V}_{V(G_i)}) = 1$ if $\overline{V}_{V(G_i)}$ satisfies $F_i$, and 0 otherwise, and $Z$ is the normalization constant, often called the *partition function*.

Next, we define junction trees. Let $\mathbb{C} = \{C_1, \ldots, C_m\}$ be a collection of subsets of $V$ such that: (a) $\cup_{i=1}^m C_i = V$ and (b) for each feature $F_j$, there exists a $C_i \in \mathbb{C}$ such that all variables of $F_j$ are contained in $C_i$. Each $C_i$ is referred to as a clique.

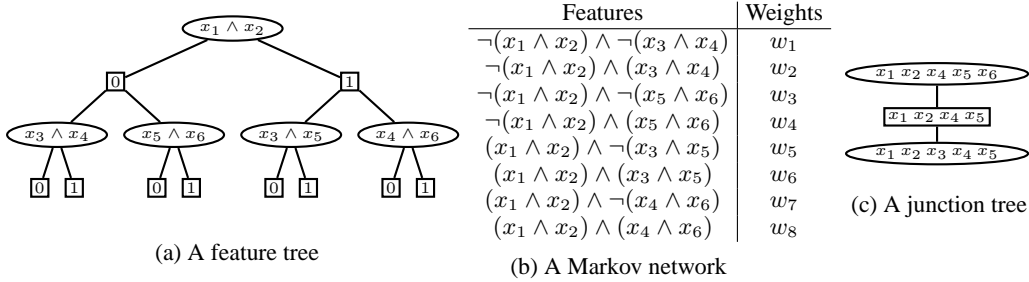

| Features | Weights |
|---|---|
| $\neg(x_1 \wedge x_2) \wedge \neg(x_3 \wedge x_4)$ | $w_1$ |
| $\neg(x_1 \wedge x_2) \wedge (x_3 \wedge x_4)$ | $w_2$ |
| $\neg(x_1 \wedge x_2) \wedge \neg(x_5 \wedge x_6)$ | $w_3$ |
| $\neg(x_1 \wedge x_2) \wedge (x_5 \wedge x_6)$ | $w_4$ |
| $(x_1 \wedge x_2) \wedge \neg(x_3 \wedge x_5)$ | $w_5$ |
| $(x_1 \wedge x_2) \wedge (x_3 \wedge x_5)$ | $w_6$ |
| $(x_1 \wedge x_2) \wedge \neg(x_4 \wedge x_6)$ | $w_7$ |
| $(x_1 \wedge x_2) \wedge (x_4 \wedge x_6)$ | $w_8$ |

(a) A feature tree    (b) A Markov network    (c) A junction tree

Figure 1: Figure showing (a) a feature tree, (b) the Markov network corresponding to the leaf features of (a) and (c) the (optimal) junction tree for the Markov network in (b). A leaf feature is formed by conjoining the feature assignments along the path from the leaf to the root. For example, the feature corresponding to the right most leaf node is: $(x_1 \wedge x_2) \wedge (x_4 \wedge x_6)$. For the feature tree, ovals denote F-nodes and rectangles denote A-nodes. For the junction tree, ovals denote cliques and rectangles denote separators. Notice that each F-node in the feature tree has a feature of size bounded by 2 while the maximum clique in the junction tree is of size 5. Moreover notice that the A-node corresponding to $(x_1 \wedge x_2) = 0$ induces a different variable decomposition as compared with the A-node corresponding to $(x_1 \wedge x_2) = 1$.

DEFINITION 1. A tree $T = (\mathbb{C}, E)$ is a **junction tree** iff it satisfies the running intersection property, i.e., $\forall C_i, C_j, C_k \in \mathbb{C}, i \neq j \neq k$, such that $C_k$ lies on the unique simple path between $C_i$ and $C_j$, $x \in C_i \cap C_j \Rightarrow x \in C_k$. The treewidth of $T$, denoted by $w$, is the size of the largest clique in $\mathbb{C}$ minus one. The set $S_{ij} \equiv C_i \cap C_j$ is referred to as the separator corresponding to the edge $(i-j) \in E$. The space complexity of representing a junction tree is $O(\sum_{i=1}^{m} 2^{|C_i|}) \equiv O(n \times 2^{w+1})$.

Our goal is to exploit context-specific and deterministic dependencies that is not explicitly represented in junction trees. Representations that do this include arithmetic circuits [10] and AND/OR graphs [11]. We will use a more convenient form for our purposes, which we call *feature graphs*. Inference in feature graphs is linear in the size of the graph. For readers familiar with AND/OR graphs [11], a feature tree (or graph) is simply an AND/OR tree (or graph) with OR nodes corresponding to features and AND nodes corresponding to feature assignments.

DEFINITION 2. A **feature tree** denoted by $S_T$ is a rooted-tree that consists of alternating levels of feature nodes or F-nodes and feature assignment nodes or A-nodes. Each F-node $\mathcal{F}$ is labeled by a feature $F$ and has two child A-nodes labeled by 0 and 1, corresponding to the true and the false assignments of $F$ respectively. Each A-node $\mathcal{A}$ has $k \geq 0$ child F-nodes that satisfy the following requirement. Let $\{\mathcal{F}_{\mathcal{A},1}, \ldots, \mathcal{F}_{\mathcal{A},k}\}$ be the set of child F-nodes of $\mathcal{A}$ and let $D(\mathcal{F}_{\mathcal{A},i})$ be the union of all variables involved in the features associated with $\mathcal{F}_{\mathcal{A},i}$ and all its descendants, then $\forall i, j \in \{1, \ldots, k\}, i \neq j, D(\mathcal{F}_{\mathcal{A},i}) \cap D(\mathcal{F}_{\mathcal{A},j}) = \emptyset$.

Semantically, each F-node represents conditioning while each A-node represents partitioning of the variables into conditionally-independent subsets. The space complexity of representing a feature tree is the number of its A-nodes. A feature graph denoted by $S_G$ is formed by merging identical subtrees of a feature tree $S_T$. It is easy to show that a feature graph generalizes a junction tree and in fact any model that can be represented using a junction tree having treewidth $k$ can also be represented by a feature graph that uses only $O(n \times 2^k)$ space [11]. In some cases, a feature graph can be exponentially smaller than a junction tree because it can capture context-specific independence [6].

A feature tree can be easily converted to a Markov network. The corresponding Markov network has one feature for each leaf node, formed by conjoining all feature assignments from the root to the leaf. The following example demonstrates the relationship between a feature tree, a Markov network and a junction tree.

EXAMPLE 1. Figure 1(a) shows a feature tree. Figure 1(b) shows the Markov network corresponding to the leaf features of the feature tree given in Figure 1(a). Figure 1(c) shows the junction tree for the Markov network given in 1(b). Notice that because the feature tree uses context-specific independence, all the $F$-nodes in the feature tree have a feature of size bounded by 2 while the maximum clique size of the junction tree is 5. The junction tree given in Figure 1(b) requires $2^5 \times 2 = 64$ potential values while the feature tree given in Figure 1(a) requires only 10 A-nodes.

In this paper, we will present structure learning algorithms to learn feature trees only. We can do this without loss of generality, because a feature graph can be constructed by caching information and merging identical nodes, while learning (constructing) a feature tree.

The distribution represented by a feature tree $S_T$ can be defined procedurally as follows (for more details see [11]). We assume that each leaf A-node $\mathcal{A}_l$ is associated with a weight $w(\mathcal{A}_l)$. For each A-node $\mathcal{A}$ and each F-node $\mathcal{F}$, we associate a value denoted by $v(\mathcal{A})$ and $v(\mathcal{F})$ respectively. We compute these values recursively as follows from the leaves to the root. The value of all A-nodes is initialized to 1 while the value of all F-nodes is initialized to 0. The value of the leaf A-node $\mathcal{A}_l$ is $w(\mathcal{A}_l) \times \#(M(\mathcal{A}_l))$ where $\#(M(\mathcal{A}_l))$ is number of (full) variable assignments that satisfy the constraint $M(\mathcal{A}_l)$ formed by conjoining the feature-assignments from the root to $\mathcal{A}_l$. The value of an internal F-node is the sum of the values of the child A-nodes. The value of an internal A-node $\mathcal{A}_p$ that has $k$ children is the product of the values of its child F-nodes divided by $[\#(M(\mathcal{A}_p))]^{k-1}$ (the division takes care of double counting). Let $v(\mathcal{F}_r)$ be the value of the root node; computed as described above. Let $\overline{V}$ be an assignment to all variables $V$ of the feature tree, then:

$$P(\overline{V}) = \frac{v_{\overline{V}}(\mathcal{F}_r)}{v(\mathcal{F}_r)}$$

where $v_{\overline{V}}(\mathcal{F}_r)$ is the value of the root node of $S_T$ computed as above in which each leaf A-node is initialized instead to $w(\mathcal{A}_l)$ if $\overline{V}$ satisfies the constraint formed by conjoining the feature-assignments from the root to $\mathcal{A}_l$ and 0 otherwise.

## 3 Learning Efficient Structure

---
**Algorithm 1**: LMIP: Low Mutual Information Partitioning

---
**Input**: A variable set $V$, sample data $D$, mutual information subroutine $I$, a feature assignment $\overline{F}$, threshold $\delta$, max set size $q$.
**Output**: A set of subsets of $V$
$\mathbb{Q}_{\overline{F}} = \{Q_1, \ldots, Q_{|V|}\}$, where $Q_i = \{x_i\}$ // $\mathbb{Q}_{\overline{F}}$ is a set of singletons
**if** *the subset of $D$ that satisfies $\overline{F}$ is too small* **then**
$\quad$ **return** $\mathbb{Q}_{\overline{F}}$
**else**
$\quad$ **for** $A \subseteq V$, $|A| \leq q$ **do**
$\quad\quad$ **if** $min_{X \subset A} I(X, A \backslash X | \overline{F}) > \delta$ **then**
$\quad\quad\quad$ // find min using Queyranne's algorithm [23] applied to the
$\quad\quad\quad\quad$ subset of $D$ satisfying $\overline{F}$
$\quad\quad\quad$ merge all $Q_i \in \mathbb{Q}_{\overline{F}}$ s.t. $Q_i \cap A \neq \emptyset$.

**return** $\mathbb{Q}_{\overline{F}}$

---

We propose a feature-based structure learning algorithm that searches for a feature that divides the configuration space into subspaces. We will assume that the selected feature or its negation divides the (remaining) variables into conditionally independent partitions (we don't require this assumption to be always satisfied, as we explain in the section on greedy heuristics and implementation details). In practice, the notion of conditional independence is too strong. Therefore, as in previous work [21, 8], we instead use conditional mutual information, denoted by $I$, to partition the set of variables. For this we use the LMIP subroutine (see Algorithm 1), a variant of Chechetka and Guestrin's [8] LTCI algorithm that outputs a partitioning of $V$. The runtime guarantees of LMIP follow from those of LTCI and correctness guarantees follow in an analogous fashion. In general, estimating mutual information between sets of random variables has time and sample complexity exponential in the number of variables considered. However, we can be more efficient as we show below. We start with a required definition.

DEFINITION 3. Given a feature assignment $\overline{F}$, a distribution $P(V)$ is $(j, \epsilon, \overline{F})$-coverable if there exists a set of cliques $\mathbb{C}$ such that for every $C_i \in \mathbb{C}$, $|C_i| \leq j$ and $I(C_i, V \setminus C_i | \overline{F}) \leq \epsilon$. Similarly, given a feature $F$, a distribution $P(V)$ is $(j, \epsilon, F)$-coverable if it is both $(j, \epsilon, F = 0)$-coverable and $(j, \epsilon, F = 1)$-coverable.

LEMMA 1. *Let $A \subset V$. Suppose there exists a distribution on $V$ that is $(j, \epsilon, \overline{F})$-coverable and $\forall X \subset V$ where $|X| \leq j$, it holds that $I(X \cap A, X \cap (V \backslash A) | \overline{F}) \leq \delta$. Then, $I(A, V \backslash A | \overline{F}) \leq |V|(2\epsilon + \delta)$.*

Lemma 1 immediately leads to the following lemma:

LEMMA **2.** *Let $P(V)$ be a distribution that is $(j, \epsilon, \overline{F})$-coverable. Then LMIP, for $q \geq j$, returns a partitioning of $V$ into disjoint subsets $\{Q_1, \ldots, Q_m\}$ such that $\forall i, I(Q_i, V \backslash Q_i | \overline{F}) \leq |V|(2\epsilon + (j - 1)\delta)$.*

We summarize the time and space complexity of LMIP in the following lemma.

LEMMA **3.** *The time and space complexity of LMIP is $O(\binom{n}{q} \times n \times J_q^{MI})$ where $J_q^{MI}$ is the time complexity of estimating the mutual information between two disjoint sets which have combined cardinality $q$.*

Note that our actual algorithm will use a subroutine that estimates mutual information from data, and the time complexity of this routine will be described in the section on sample complexity and probabilistic performance guarantees.

---

**Algorithm 2**: LEM: Learning Efficient Markov Networks

**Input**: Variable set $V$, sample data $S$, mutual information subroutine $I$, feature length $k$, set size parameter $q$, threshold $\delta$, an A-node $\mathcal{A}$.
**Output**: A feature tree $\mathcal{M}$
**for** *each feature $F$ of length $k$ constructible for $V$* **do**
  $\mathbb{Q}_{F=1} = $ LMIP($V$, $S$, $I$, $F = 1$, $\delta$, $q$);
  $\mathbb{Q}_{F=0} = $ LMIP ($V$, $S$, $I$, $F = 0$, $\delta$, $q$)
$G = argmax_F$ (Score($\mathbb{Q}_{F=0}$)+ Score($\mathbb{Q}_{F=1}$))// $G$ is a feature
**if** $|\mathbb{Q}_{G=0}| = 1$ *and* $|\mathbb{Q}_{G=1}| = 1$ **then**
  Create a feature tree corresponding to all possible assignments to the atomic features. Add this feature tree as a child of $\mathcal{A}$;
  **return**
Create a F-node $\mathcal{G}$ with $G$ as its feature, and add it as a child of $\mathcal{A}$;
Create two A-child nodes $\mathcal{A}_{\mathcal{G},0}$ and $\mathcal{A}_{\mathcal{G},1}$ for $\mathcal{G}$;
**for** $i \in \{0, 1\}$ **do**
  **if** $|\mathbb{Q}_{G=i}| > 1$ **then**
    **for** *each component (subset of $V$) $C \in \mathbb{Q}_{G=i}$* **do**
      $S_C = \text{Project}_C(\{X \in S : X \text{ satisfies } G = i\})$ // $S_C$ is the set of instantiations of $V$ in $S$ that satisfy $G = i$ restricted to the variables in $C$
      LEM($C$, $S_C$, $I$, $k$, $q$, $\delta$, $\mathcal{A}_{\mathcal{G},i}$) // Recursion
  **else**
    Create a feature tree corresponding to all possible assignments to the atomic features. Add this feature tree as a child of $\mathcal{A}_{\mathcal{G},i}$.

---

Next, we present our structure learning algorithm called LEM (see Algorithm 2) which utilizes the LMIP subroutine to learn feature trees from data. The algorithm has probabilistic performance guarantees if we make some assumptions on the type of the distribution. We present these guarantees in the next subsection. Algorithm 2 operates as follows. First, it runs the LMIP subroutine on all possible features of length $k$ constructible from $V$. Recall that given a feature assignment $\overline{F}$, the LMIP sub-routine partitions the variables into (approximately) conditionally independent components. It then selects a feature $G$ having the highest score. Intuitively, to reduce the inference time and the size of the model, we should try to balance the trade-off between increasing the number of partitions and maintaining partition size uniformity (namely, we would want the partition sizes to be almost equal). The following score function achieves this objective. Let $\mathbb{Q} = \{Q_1, \ldots, Q_m\}$ be a m-partition of $V$, then the **score** of $\mathbb{Q}$ is given by: $Score(\mathbb{Q}) = \frac{1}{\sum_{i=1}^{m} 2^{|Q_i|}}$, where the denominator bounds worst-case inference complexity.

After selecting a feature $G$, the algorithm creates a F-node corresponding to $G$ and two child A-nodes corresponding to the true and the false assignments of $G$. Then, corresponding to each element of $\mathbb{Q}_{G=1}$, it recursively creates a child node for $G = 1$ (and similarly for $G = 0$ using $\mathbb{Q}_{G=0}$). An interesting special case is when either $|\mathbb{Q}_{G=1}| = 1$ or $|\mathbb{Q}_{G=0}| = 1$ or when both conditions hold. In this case, no partitioning of $V$ exists for either or both the value assignments of $G$ and therefore we return a feature tree which has $2^{|V|}$ leaf $A$-nodes corresponding to all possible instantiations of the remaining variables. In practice, because of the exponential dependence on $|V|$, we would want

this condition to hold only when a few variables remain. To obtain guarantees, however, we need stronger conditions to be satisfied. We describe these guarantees next.

## 3.1 Theoretical Guarantees

To derive performance guarantees and to guarantee polynomial complexity, we make some fundamental assumptions about the data and the distribution $P(V)$ that we are trying to learn. Intuitively, if there exists a feature $F$ such that the distribution $P(V)$ at each recursive call to LEM is $(j, \epsilon, F)$-coverable, then the LMIP sub-routine is guaranteed to return at least a two-way partitioning of $V$. Assume that $P(V)$ is such that at each recursive call to LEM, there exists a unique $F$ (such that the distribution at the recursive call is $(j, \epsilon, F)$-coverable). Then, LEM is guaranteed to find this unique feature tree. However, the trouble is that at each step of the recursion, there may exist $m > 1$ *candidate features* that satisfy this property. Therefore, we want this coverability requirement to hold not only recursively but also for each candidate feature (at each recursive call). The following two definitions and Theorem 1 capture this intuition.

DEFINITION 4. Given a constant $\delta > 0$, we say that a distribution $P(V)$ satisfies the $(j, \epsilon, m, \overline{G})$ assumption if $|V| \leq j$ or if the following property is satisfied. For every feature $F$, and each assignment $\overline{F}$ of $F$, such that $|V(F)| \leq m$, $P(V)$ is $(j, \epsilon, \overline{F})$-coverable and for any partitioning $S_1, ..., S_z$ of $V$ with $z \geq 2$, such that for each $i$, $I(S_i, V \setminus S_i | \overline{F} \wedge \overline{G}) \leq |V|(2\epsilon + \delta)$ and $P(S_1), ..., P(S_z)$ each satisfy the $(j, \epsilon, m, \overline{G} \wedge \overline{F})$ assumption.

DEFINITION 5. We say the a sequence of pairs $(\overline{F}_n, S_n), (\overline{F}_{n-1}, S_{n-1}), \ldots, (\overline{F}_0, S_0 = V)$ satisfies the **nested context independence condition for** $(\theta, w)$ if $\forall i, S_i \subseteq S_{i-1}$ and the distribution on $V$ conditioned on the satisfaction of $\overline{G}_{i-1} = (\overline{F}_{i-1} \wedge \overline{F}_{i-2} \wedge \ldots \wedge \overline{F}_0)$ is such that $I(S_i, S_{i-1} \setminus S_i | \overline{G}_{i-1}) \leq |S_{i-1}|(2\theta + w)$.

THEOREM 1. *Given a distribution $P(V)$ that satisfies the $(j, \epsilon, m, true)$-assumption and a perfect mutual information oracle $I$, LEM(V, S, I, k, j + 1, $\delta$) returns a feature tree $S_T$ such that each leaf feature of $S_T$ satisfies the nested context independence condition for $(\epsilon, j \times \delta)$.*

### 3.1.1 Sample Complexity and Probabilistic Performance Guarantees

The foregoing analysis relies on a perfect, deterministic mutual information subroutine $I$. In reality, all we have is sample data and probabilistic mutual information subroutines. As the following theorem shows, we can get estimates of $I(A, B | \overline{F})$ with accuracy $\pm\Delta$ and probability $1 - \gamma$ with a number of samples and running time polynomial in $\frac{1}{\Delta}$ and $\log \frac{1}{\gamma}$.

LEMMA 4. *(Hoffgen [14]) The entropy of a probability distribution over $2k + 2$ discrete variables with domain size $R$ can be estimated with accuracy $\Delta$ with probability at least $1 - \gamma$ using $F(k, R, \Delta, \gamma) = O(\frac{R^{4k+4}}{\Delta^2} log^2(\frac{R^{2k+2}}{\Delta^2}) log(\frac{R^{2k+2}}{\gamma}))$ samples and the same amount of time.*

To ensure that our algorithm doesn't run out of data somewhere in the recursion, we have to strengthen our assumptions, as we define below.

DEFINITION 6. If $P(V)$ satisfies the $(j, \epsilon, m, true)$-assumption and a set of sample data $H$ drawn from the distribution is such that for any $\overline{G}_{i-1} = \overline{F}_{i-1} \wedge \ldots \overline{F}_0$ if neither $F_i = 0$ or $F_i = 1$ hold in less than some constant fraction $c$ of the subset of $H$ that satisfies $\overline{G}_{i-1}$, then we say that $H$ satisfies the $c$-**strengthened** $(j, \epsilon, m, true)$ **assumption**.

THEOREM 2 (**Probabilistic performance guarantees**). *Let $P(V)$ be a distribution that satisfies the $(j, \epsilon, m, true)$ assumption and let $H$ be the training data which satisfies the $c$-strengthened $(j, \epsilon, m, true)$ assumption from which we draw $S$ samples of size $T = (\frac{1}{c})^D F(\frac{j-1}{2}, |V|, \Delta, \frac{\gamma}{n^{m+j+2}(j+1)^3})$, where $D$ is the worst-case length of any leaf feature returned by the algorithm. Given a mutual information subroutine $\hat{I}$ implied by Lemma 4, LEM(V, S, $\hat{I}$, m, j + 1, $\epsilon + \Delta$) returns a feature tree, the leaves of which satisfy the nested context independence condition for $(\epsilon, j \times (\epsilon + \Delta))$, with probability $1 - \gamma$.*

## 4 Greedy Heuristics and Implementation Details

When implemented naively, Algorithm 2 may be computationally infeasible. The most expensive step in LEM is the LMIP sub-routine which is called $O(n^k)$ times at each A-node of the feature

graph. Given a max set size of $q$, LMIP requires running Queyranne's algorithm [23] (complexity $O(q^3)$) to minimize $min_{X \subset A} I(X, V \setminus X | \overline{F})$ over every $|A| \leq q$. Thus, its overall time complexity is $O(n^q \times q^3)$. Also, our theoretical analysis assumes access to a mutual information oracle which is not available in practice and one has to compute $I(X, V \setminus X | \overline{F})$ from data. In our implementation, we used Moore and Lee's AD-trees [19] to pre-compute and cache the sufficient statistics (counts), in advance, so that at each step, $I(X, V \setminus X | \overline{F})$ can be computed efficiently. A second improvement that we considered is due to Chechtka and Guestrin [8]. It is based on the observation that if $A$ is a subset of a connected component $Q \in \mathbb{Q}_{\overline{F}}$, then we don't need to compute $min_{X \subset A} I(X, V \setminus X | \overline{F})$, because merging all $Q_i \in \mathbb{Q}_{\overline{F}}$ s.t. $Q_i \cap A \neq \emptyset$. would not change $\mathbb{Q}_{\overline{F}}$. In spite of these improvements, our algorithm is not practical for $q > 3$ and $k > 3$. Note however, that low values of $q$ and $k$ are not entirely problematic for our approach because we may still be able to induce large treewidth models by taking advantage of context specific independence, as depicted in Figure 1.

To further improve the performance of our algorithm, we fix $q$ to 3 and use a greedy heuristic to construct the features. The greedy heuristic is able to split on arbitrarily long features by only calling LMIP $k \times n$ times instead of $O(n^k)$ times, but does not have any guarantees. It starts with a set of atomic features (i.e., just the variables in the domain), runs LMIP on each, and selects the (best) feature with the highest score. Then, it creates candidate features by conjoining this best feature from the previous step with each atomic feature, runs LMIP on each, and then selects a best feature for the next iteration. It repeats this process until $i$ equals $k$ or the score does not improve. This heuristic is loosely based on the greedy approach of Della Pietra et al.[12]. We also use a *balance heuristic* to reduce the size of the model learned; which imposes a form of regularization constraint and biases our search towards sparser models, in order to avoid over-fitting. Here, given a set of features with similar scores, we select a feature $F$ such that the difference between the scores of $F = 0$ and $F = 1$ is the smallest. The intuition behind this heuristic is that by maintaining balance we reduce the height of the feature graph and thus its size. Finally, in our implementation, we do not return all possible instantiations of the variables when a feature assignment yields only one partition, unless the number of remaining variables is smaller than 5. This is because even though a feature may not partition the set of variables, it may still partition the data, thereby reducing complexity.

## 5   Experimental Evaluation

We evaluated LEM on one synthetic data set and four real world ones. Figure 2(f) lists the five data sets and the number of atomic features in each. The synthetic domain consists of samples from the Alarm Bayesian network [3]. From the UCI machine learning repository [5], we used the Adult and MSNBC anonymous Web data domains. Temperature and Traffic are sensor network data sets and were used in Checketka and Guestrin [8].

We compared LEM to the standard Markov network structure learning algorithm of Della Pietra et al.[12] (henceforth, called the DL scheme), the L1 approach of Ravikumar et al. [24] and the lazy thin-junction tree algorithm (LPACJT) of Chechetka and Guestrin [8]. We used the following parameters for LEM: $q = 3$, and $\delta = 0.05$. We found that the results were insensitive to the value of $\delta$ used. We suggest using any reasonably small value $\leq 0.1$. The LPACJT implementation available from the authors requires entropies (computed from the data) as input. We were unable to compute the entropies in the required format because they use a propriety software that we did not have access to, and therefore we use the results provided by the authors for the temperature, traffic and alarm domains. We were unable to run LPACJT on the other two domains. We altered the DL algorithm to only evaluate candidate features that match at least one example. This simple extension vastly reduces the number of candidate features and greatly improves the algorithm's efficiency. For implementing DL, we use pseudo-likelihood [4] as a scoring function and optimized it via the limited-memory BFGS algorithm [17]. For implementing L1, we used the OWL-QN software package of Andrew and Gao [1]. The neighborhood structures for L1 can be merged in two ways (logical-OR or logical-AND of the structures); we tried both and used the best one for plotting the results. For the regularization, we tried penalty = $\{1, 2, 5, 10, 20, 25, 50, 100, 200, 500, 1000\}$ and used a tuning set to pick the one that gave the best results. We used a time-bound of 24 hrs for each algorithm.

For each domain, we evaluated the algorithms on training set sizes varying from 100 to 10000. We performed a five-fold train-test split. For the sensor networks, traffic and alarm domains, we use the test set sizes provided in Chechtka and Guestrin [8]. For the MSNBC and Adult domains, we selected a test set consisting of 58265 and 7327 examples respectively. We evaluate the performance

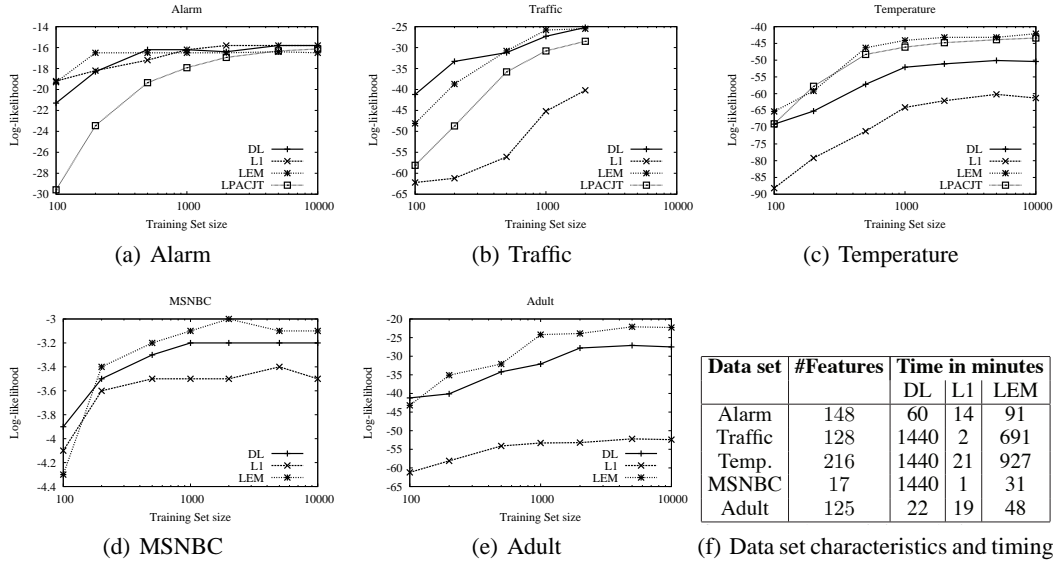

| | | | (f) Data set characteristics and timing results |

Figure 2: Figures (a)-(e) showing average log-Likelihood as a function of the training data size for LEM, DL, L1 and LPACJT. Figure (f) reports the run-time in minutes for LEM, DL and L1 for training set of size 10000.

based on average-log-likelihood of the test data, given the learned model. The log-likelihood of the test data was computed exactly for the models output by LPACJT and LEM, because inference is tractable in these models. The size of the feature graphs learned by LEM ranged from $O(n^2)$ to $O(n^3)$, comparable to those generated by LPACJT. Exact inference on the learned feature graphs was a matter of milliseconds. For the Markov networks output by DL and L1, we compute the log-likelihood approximately using loopy Belief propagation [20].

Figure 2 summarizes the results for the five domains. LEM significantly outperforms L1 on all the domains except the Alarm dataset. It is better than the greedy DL scheme on three out of the five domains while it is always better than LPACJT. Figure 2(f) shows the timing results for LEM, DL and L1. L1 is substantially faster than DL and LEM. DL is the slowest scheme.

# 6 Conclusions

We have presented an algorithm for learning a class of high-treewidth Markov networks that admit tractable inference and closed-form parameter learning. This class is much richer than thin junction trees because it exploits context-specific independence and determinism. We showed that our algorithm has probabilistic performance guarantees under the recursive assumption that the distribution at each node in the (rooted) feature graph (which is defined only over a decreasing subset of variables as we move further away from the root), is itself representable by a polynomial-sized feature graph and in which the maximum feature-size at each node is bounded by $k$. We believe that our new theoretical insights further the understanding of structure learning in Markov networks, especially those having high treewidth. In addition to the theoretical guarantees, we showed that our algorithm has good performance in practice, usually having higher test-set likelihood than other competing approaches. Although learning may be slow, inference always has quick and predictable runtime, which is linear in the size of the feature graph. Intuitively, our method seems likely to perform well on large sparsely dependent datasets.

### Acknowledgements

This research was partly funded by ARO grant W911NF-08-1-0242, AFRL contract FA8750-09-C-0181, DARPA contracts FA8750-05-2-0283, FA8750-07-D-0185, HR0011-06-C-0025, HR0011-07-C-0060 and NBCH-D030010, NSF grants IIS-0534881 and IIS-0803481, and ONR grant N00014-08-1-0670. The views and conclusions contained in this document are those of the authors and should not be interpreted as necessarily representing the official policies, either expressed or implied, of ARO, DARPA, NSF, ONR, or the United States Government.

# References

[1] G. Andrew and J. Gao. Scalable training of L1-regularized log-linear models. In *Proceedings of the Twenty-Fourth International Conference (ICML)*, pages 33–40, 2007.

[2] F. R. Bach and M. I. Jordan. Thin junction trees. In *Advances in Neural Information Processing Systems*, pages 569–576, 2001.

[3] I. Beinlich, J. Suermondt, M. Chavez, and G. Cooper. The alarm monitoring system: A case study with two probablistic inference techniques for belief networks. In *European Conference on AI in Medicine*, 1988.

[4] J. Besag. Statistical analysis of non-lattice data. *The Statistician*, 24:179–195, 1975.

[5] C. Blake and C. J. Merz. UCI repository of machine learning databases. Machine-readable data repository, Department of Information and Computer Science, University of California at Irvine, Irvine, CA, 2000. http://www.ics.uci.edu/~mlearn/MLRepository.html.

[6] C. Boutilier. Context-specific independence in Bayesian networks. In *Proceedings of the Twelfth Annual Conference on Uncertainty in Artificial Intelligence (UAI)*, pages 115–123, 1996.

[7] M. Chavira and A. Darwiche. On probabilistic inference by weighted model counting. *Artificial Intelligence*, 172(6–7):772–799, April 2008.

[8] A. Chechetka and C. Guestrin. Efficient principled learning of thin junction trees. In *Advances in Neural Information Processing Systems (NIPS)*, December 2007.

[9] D.M. Chickering, D. Geiger, and D. Heckerman. Learning Bayesian networks: Search methods and experimental results. In *Proceedings of the Fifth International Workshop on Artificial Intelligence and Statistics (AISTATS)*, pages 112–128, 1995.

[10] A. Darwiche. A differential approach to inference in Bayesian networks. *Journal of the ACM*, 50(3):280–305, 2003.

[11] R. Dechter and R. Mateescu. AND/OR search spaces for graphical models. *Artificial Intelligence*, 171(2-3):73–106, 2007.

[12] S. Della Pietra, V. Della Pietra, and J. Lafferty. Inducing features of random fields. *IEEE Transactions on Pattern Analysis and Machine Intelligence*, 19:380–392, 1997.

[13] S. Goldman. Efficient methods for calculating maximum entropy distributions. Master's thesis, Massachusetts Institute of Technology, 1987.

[14] K. Höffgen. Learning and robust learning of product distributions. In *Proceedings of the Sixth Annual ACM Conference on Computational Learning Theory (COLT)*, pages 77–83, 1993.

[15] D. R. Karger and N. Srebro. Learning Markov networks: maximum bounded tree-width graphs. In *Proceedings of the Seventeenth Annual ACM-SIAM Symposium on Discrete Algorithms (SODA)*, pages 392–401, 2001.

[16] S. Lee, V. Ganapathi, and D. Koller. Efficient structure learning of Markov networks using L1-regularization. In *Proceedings of the Twentieth Annual Conference on Neural Information Processing Systems (NIPS)*, pages 817–824, 2006.

[17] D. C. Liu and J. Nocedal. On the limited memory BFGS method for large scale optimization. *Mathematical Programming*, 45(3):503–528, 1989.

[18] D. Lowd and P. Domingos. Learning arithmetic circuits. In *Proceedings of the Twenty Fourth Conference in Uncertainty in Artificial Intelligence*, pages 383–392, 2008.

[19] A. W. Moore and M. S. Lee. Cached sufficient statistics for efficient machine learning with large datasets. *Journal of Artificial Intelligence Research*, 8:67–91, 1997.

[20] K. P. Murphy, Y. Weiss, and M. I. Jordan. Loopy belief propagation for approximate inference: An empirical study. In *Proceedings of the Fifteenth Conference on Uncertainty in Artificial Intelligence (UAI)*, pages 467–475, 1999.

[21] M. Narasimhan and J. Bilmes. PAC-learning bounded tree-width graphical models. In *Proceedings of the Twentieth Conference in Uncertainty in Artificial Intelligence (UAI)*, pages 410–417, 2004.

[22] J. Pearl. *Probabilistic Reasoning in Intelligent Systems: Networks of Plausible Inference*. Morgan Kaufmann, San Francisco, CA, 1988.

[23] M. Queyranne. Minimizing symmetric submodular functions. *Mathematical Programming*, 82(1):3–12, 1998.

[24] P. Ravikumar, M. J. Wainwright, and J. Lafferty. High-dimensional Ising model selection using L1-regularized logistic regression. *Annals of Statistics*, 38(3):1287–1319, 2010.

[25] D. Roth. On the hardness of approximate reasoning. *Artificial Intelligence*, 82:273–302, 1996.

[26] T. Sang, P. Beame, and H. Kautz. Performing Bayesian inference by weighted model counting. In *Proceedings of The Twentieth National Conference on Artificial Intelligence (AAAI)*, pages 475–482, 2005.

